# Task and Spatial Frequency Effects on Face Specialization

**Matthew N. Dailey    Garrison W. Cottrell**
Department of Computer Science and Engineering
U.C. San Diego
La Jolla, CA 92093-0114
{mdailey,gary}@cs.ucsd.edu

## Abstract

There is strong evidence that face processing is localized in the brain. The double dissociation between prosopagnosia, a face recognition deficit occurring after brain damage, and visual object agnosia, difficulty recognizing other kinds of complex objects, indicates that face and non-face object recognition may be served by partially independent mechanisms in the brain. Is neural specialization innate or learned? We suggest that this specialization could be the result of a competitive learning mechanism that, during development, devotes neural resources to the tasks they are best at performing. Further, we suggest that the specialization arises as an interaction between task requirements and developmental constraints. In this paper, we present a feed-forward computational model of visual processing, in which two modules compete to classify input stimuli. When one module receives low spatial frequency information and the other receives high spatial frequency information, and the task is to identify the faces while simply classifying the objects, the low frequency network shows a strong specialization for faces. No other combination of tasks and inputs shows this strong specialization. We take these results as support for the idea that an innately-specified face processing module is unnecessary.

## 1 Background

Studies of the preserved and impaired abilities in brain damaged patients provide important clues on how the brain is organized. Cases of prosopagnosia, a face recognition deficit often sparing recognition of non-face objects, and visual object agnosia, an object recognition deficit that can occur without appreciable impairment of face recognition, provide evidence that face recognition is served by a "special" mechanism. (For a recent review of this

evidence, see Moscovitch, Winocur, and Behrmann (1997)). In this study, we begin to provide a computational account of the double dissociation.

Evidence indicates that face recognition is based primarily on holistic, configural information, whereas non-face object recognition relies more heavily on local features and analysis of the parts of an object (Farah, 1991; Tanaka and Sengco, 1997). For instance, the distance between the tip of the nose and an eye in a face is an important factor in face recognition, but such subtle measurements are rarely as critical for distinguishing, say, two buildings. There is also evidence that configural information is highly relevant when a human becomes an "expert" at identifying individuals within other visually homogeneous object classes (Gauthier and Tarr, 1997).

What role might configural information play in the development of a specialization for face recognition? de Schonen and Mancini (1995) have proposed that several factors, including different rates of maturation in different areas of cortex, an infant's tendency to track the faces in its environment, and the gradual increase in visual acuity as an infant develops, all combine to force an early specialization for face recognition. If this scenario is correct, the infant begins to form configural face representations very soon after birth, based primarily on the low spatial frequency information present in face stimuli. Indeed, Costen, Parker, and Craw (1996) showed that although both high-pass and low-pass image filtering decrease face recognition accuracy, high-pass filtering degrades identification accuracy more quickly than low-pass filtering. Furthermore, Schyns and Oliva (1997) have shown that when asked to recognize the identity of the "face" in a briefly-presented hybrid image containing a low-pass filtered image of one individual's face and a high-pass filtered image of another individual's face, subjects consistently use the low-frequency component of the image for the task. This work indicates that low spatial frequency information may be more important for face identification than high spatial frequency information.

Jacobs and Kosslyn (1994) showed how differential availability of large and small receptive field sizes in a mixture of experts network (Jacobs, Jordan, Nowlan, and Hinton, 1991) can lead to experts that specialize for "what" and "where" tasks. In previous work, we proposed that a neural mechanism allocating resources according to their ability to perform a given task could explain the apparent specialization for face recognition evidenced by prosopagnosia (Dailey, Cottrell, and Padgett, 1997). We showed that a model based on the mixture of experts architecture, in which a gating network implements competitive learning between two simple homogeneous modules, could develop a specialization such that damage to one module disproportionately impaired face recognition compared to non-face object recognition.

In the current study, we consider how the availability of spatial frequency information affects face recognition specialization given this hypothesis of neural resource allocation by competitive learning. We find that when high and low frequency information is "split" between the two modules in our system, and the task is to identify the faces while simply classifying the objects, the low-frequency module consistently specializes for face recognition. After describing the study, we discuss its results and their implications.

## 2   Experimental Methods

We presented a modular feed-forward neural network preprocessed images of 12 different faces, 12 different books, 12 different cups, and 12 different soda cans. We gave the network two types of tasks:

1. Learning to recognize the superordinate classes of all four object types (hereafter referred to as *classification*).

2. Learning to distinguish the individual members of one class (hereafter referred to

as *identification*) while simply classifying objects of the other three types.

For each task, we investigated the effects of high and low spatial frequency information on identification and classification in a visual processing system with two competing modules. We observed how splitting the range of spatial frequency information between the two modules affected the specializations developed by the network.

## 2.1 Image Data

We acquired face images from the Cottrell and Metcalfe facial expression database (1991) and captured multiple images of several books, cups, and soda cans with a CCD camera and video frame grabber. For the face images, we chose five grayscale images of each of 12 individuals. The images were photographed under controlled lighting and pose conditions; the subjects portrayed a different facial expression in each image. For each of the non-face object classes, we captured five different grayscale images of each of 12 books, 12 cups, and 12 cans. These images were also captured under controlled lighting conditions, with small variations in position and orientation between photos. The entire image set contained 240 images, each of which we cropped and scaled to a size of 64x64 pixels.

## 2.2 Image Preprocessing

To convert the raw grayscale images to a biologically plausible representation more suitable for network learning and generalization, and to experiment with the effect of high and low spatial frequency information available in a stimulus, we extracted Gabor jet features from the images at multiple spatial frequency scales then performed a separate principal components analysis on the data from each filter scale separately to reduce input pattern dimensionality.

### 2.2.1 Gabor jet features

The basic two-dimensional Gabor wavelet resembles a sinusoid grating restricted by a two-dimensional Gaussian, and may be tuned to a particular orientation and sinusoidal frequency scale. The wavelet can be used to model simple cell receptive fields in cat primary visual cortex (Jones and Palmer, 1987). Buhmann, Lades, and von der Malsburg (1990) describe the Gabor "jet," a vector consisting of filter responses at multiple orientations and scales.

We convolved each of the 240 images in the input data set with two-dimensional Gabor filters at five scales in eight orientations $(0, \frac{\pi}{8}, \frac{\pi}{4}, \frac{3\pi}{8}, \frac{\pi}{2}, \frac{5\pi}{8}, \frac{3\pi}{4}, \frac{7\pi}{8})$ and subsampled an 8x8 grid of the responses to each filter. The process resulted in 2560 complex numbers describing each image.

### 2.2.2 Principal components analysis

To reduce the dimensionality of the Gabor jet representation while maintaining a segregation of the responses from each filter scale, we performed a separate PCA on each spatial frequency component of the pattern vector described above. For each of the 5 filter scales in the jet, we extracted the subvectors corresponding to that scale from each pattern in the training set, computed the eigenvectors of their covariance matrix, projected the subvectors from each of the patterns onto these eigenvectors, and retained the eight most significant coefficients. Reassembling the pattern set resulted in 240 40-dimensional vectors.

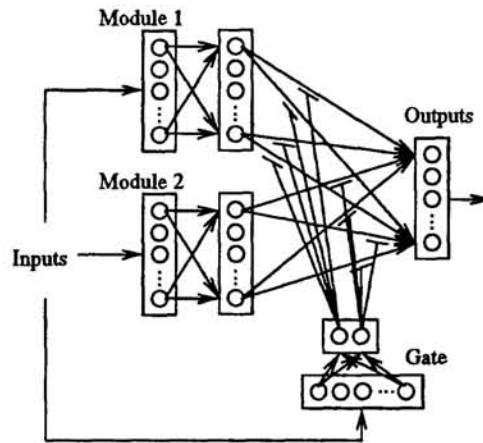

Figure 1: Modular network architecture. The gating network units mix the outputs of the hidden layers multiplicatively.

## 2.3 The Model

The model is a simple modular feed-forward network inspired by the mixture of experts architecture (Jordan and Jacobs, 1995); however, it contains hidden layers and is trained by backpropagation of error rather than maximum likelihood estimation or expectation maximization. The connections to the output units come from two separate input/hidden layer pairs; these connections are gated multiplicatively by a simple linear network with softmax outputs. Figure 1 illustrates the model's architecture. During training, the network's weights are adjusted by backpropagation of error. The connections from the softmax units in the gating network to the connections between the hidden layers and output layer can be thought of as multiplicative connections with a constant weight of 1. The resulting learning rules gate the amount of error feedback received by a module according to the gating network's current estimate of its ability to process the current training pattern. Thus the model implements a form of competitive learning in which the gating network learns which module is better able to process a given pattern and rewards the "winner" with more error feedback.

## 2.4 Training Procedure

Preprocessing the images resulted in 240 40-dimensional vectors; four examples of each face and object composed a 192-element training set, and one example of each face and object composed a 48-element test set. We held out one example of each individual in the training set for use in determining when to stop network training. We set the learning rate for all network weights to 0.1 and their momentum to 0.5. Both of the hidden layers contained 15 units in all experiments. For the identification tasks, we determined that a mean squared error (MSE) threshold of 0.02 provided adequate classification performance on the hold out set without overtraining and allowed the gate network to settle to stable values. For the four-way classification task, we found that an MSE threshold of 0.002 was necessary to give the gate network time to stabilize and did not result in overtraining. On all runs reported in the results section, we simply trained the network until it reached the relevant MSE threshold.

For each of the tasks reported in the results section (four-way classification, book identification, and face identification), we performed two experiments. In the first, as a control, both modules and the gating network were trained and tested with the full 40-dimensional pattern vector. In the second, the gating network received the full 40-dimensional vector,

but module 1 received a vector in which the elements corresponding to the largest two Gabor filter scales were set to 0, and the elements corresponding to the middle filter scale were reduced by 0.5. Module 2, on the other hand, received a vector in which the elements corresponding to the smallest two filter scales were set to 0 and the elements corresponding to the middle filter were reduced by 0.5. Thus module 1 received mostly high-frequency information, whereas module 2 received mostly low-frequency information, with deemphasized overlap in the middle range.

For each of these six experiments, we trained the network using 20 different initial random weight sets and recorded the softmax outputs learned by the gating network on each training pattern.

## 3 Results

Figure 2 displays the resulting degree of specialization of each module on each stimulus class. Each chart plots the average weight the gating network assigns to each module for the training patterns from each stimulus class, averaged over 20 training runs with different initial random weights. The error bars denote standard error. For each of the three reported tasks (four-way classification, book identification, and face identification), one chart shows division of labor between the two modules in the control situation, in which both modules receive the same patterns, and the other chart shows division of labor between the two modules when one module receives low-frequency information and the other receives high-frequency information.

When required to identify faces on the basis of high- or low-frequency information, compared with the four-way-classification and same-pattern controls, the low-frequency module wins the competition for face patterns extremely consistently (lower right graph). Book identification specialization, however, shows considerably less sensitivity to spatial frequency.

We have also performed the equivalent experiments with a cup discrimination and a can discrimination task. Both of these tasks show a low-frequency sensitivity lower than that for face identification but higher than that for book identification. Due to space limitations, these results are not presented here.

The specialized face identification networks also provide good models of prosopagnosia and visual object agnosia: when the face-specialized module's output is "damaged" by removing connections from its hidden layer to the output layer, the overall network's generalization performance on face identification drops dramatically, while its generalization performance on object recognition drops much more slowly. When the non-face-specialized (high frequency) module's outputs are damaged, the opposite effect occurs: the overall network's performance on each of the object recognition tasks drops, whereas its performance on face identification remains high.

## 4 Discussion

The results in Figure 2 show a strong preference for low-frequency information in the face identification task, empirically demonstrating that, given a choice, a competitive mechanism will choose a module receiving low-frequency, large receptive field information for this task. This result concurs with the psychological evidence for configural face representations based upon low spatial frequency information, and suggests how the developing brain could be biased toward a specialization for face recognition by the infant's initially low visual acuity.

On the basis of these results, we predict that human subjects performing face and object

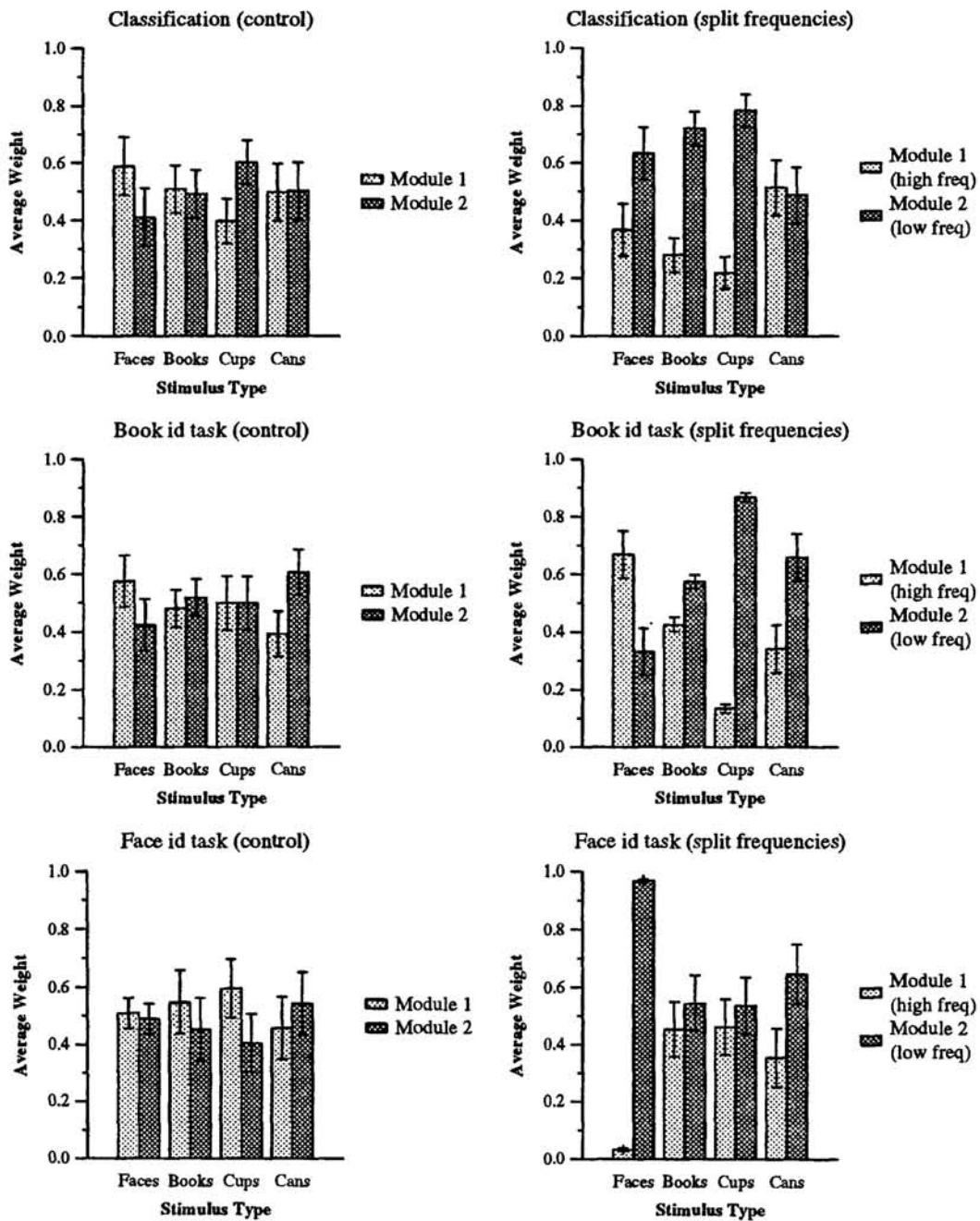

Figure 2: Average weight assigned to each module broken down by stimulus class. For each task, in the control experiment, each module receives the same pattern; the split-frequency charts summarize the specialization resulting when module 1 receives high-frequency Gabor filter information and module 2 receives low-frequency Gabor filter information.

identification tasks will show more degradation of performance in high-pass filtered images of faces than in high-pass filtered images of other objects. To our knowledge, this has not been empirically tested, although Costen et al. (1996) have investigated the effect of high-pass and low-pass filtering on face images in isolation, and Parker, Lishman, and Hughes (1996) have investigated the effect of high-pass and low-pass filtering of face and object images used as 100 ms cues for a same/different task. Their results indicate that relevant high-pass filtered images cue object processing better than low-pass filtered images, but the two types of filtering cue face processing equally well. Similarly, Schyns & Oliva's (1997) results described earlier suggest that the human face identification network preferentially responds to low spatial frequency inputs.

Our results suggest that simple data-driven competitive learning combined with constraints and biases known or thought to exist during visual system development can account for some of the effects observed in normal and brain-damaged humans. The study lends support to the claim that there is no need for an innately-specified face processing module — face recognition is only "special" insofar as faces form a remarkably homogeneous category of stimuli for which within-category discrimination is ecologically beneficial.

## References

Buhmann, J., Lades, M., and von der Malsburg, C. (1990). Size and distortion invariant object recognition by hierarchical graph matching. In *Proceedings of the IJCNN International Joint Conference on Neural Networks*, volume II, pages 411–416.

Costen, N., Parker, D., and Craw, I. (1996). Effects of high-pass and low-pass spatial filtering on face identification. *Perception & Psychophysics*, 38(4):602–612.

Cottrell, G. and Metcalfe, J. (1991). Empath: Face, gender and emotion recognition using holons. In Lippman, R., Moody, J., and Touretzky, D., editors, *Advances in Neural Information Processing Systems 3*, pages 564–571.

Dailey, M., Cottrell, G., and Padgett, C. (1997). A mixture of experts model exhibiting prosopagnosia. In *Proceedings of the Nineteenth Annual Conference of the Cognitive Science Society*, pp. 155-160. Stanford, CA, Mahwah: Lawrence Erlbaum.

de Schonen, S. and Mancini, J. (1995). About functional brain specialization: The development of face recognition. TR 95.1, MRC Cognitive Development Unit, London, UK.

Farah, M. (1991). Patterns of co-occurrence among the associative agnosias: Implications for visual object representation. *Cognitive Neuropsychology*, 8:1–19.

Gauthier, I. and Tarr, M. (1997). Becoming a "greeble" expert: Exploring mechanisms for face recognition. *Vision Research*. In press.

Jacobs, R. and Kosslyn, S. (1994). Encoding shape and spatial relations — The role of receptive field size in coordinating complementary representations. *Cognitive Science*, 18(3):361–386.

Jacobs, R., Jordan, M., Nowlan, S., and Hinton, G. (1991). Adaptive mixtures of local experts. *Neural Computation*, 3:79–87.

Jones, J. and Palmer, L. (1987). An evaluation of the two-dimensional Gabor filter model of simple receptive fields in cat striate cortex. *J. Neurophys.*, 58(6):1233–1258.

Moscovitch, M., Winocur, G., and Behrmann, M. (1997). What is special about face recognition? Nineteen experiments on a person with visual object agnosia and dyslexia but normal face recognition. *Journal of Cognitive Neuroscience*, 9(5):555–604.

Parker, D., Lishman, J., and Hughes, J. (1996). Role of coarse and fine spatial information in face and object processing. *Journal of Experimental Psychology: Human Perception and Performance*, 22(6):1445–1466.

Schyns, P. and Oliva, A. (1997). Dr. Angry and Mr. Smile: The multiple faces of perceptual categorizations. Submitted for publication.

Tanaka, J. and Sengco, J. (1997). Features and their configuration in face recognition. *Memory and Cognition*. In press.